# A Variational Approach to Learning Curves

**Dörthe Malzahn**    **Manfred Opper**
Neural Computing Research Group
School of Engineering and Applied Science
Aston University, Birmingham B4 7ET, United Kingdom.
`[malzahnd,opperm]@aston.ac.uk`

## Abstract

We combine the replica approach from statistical physics with a variational approach to analyze learning curves analytically. We apply the method to Gaussian process regression. As a main result we derive approximative relations between empirical error measures, the generalization error and the posterior variance.

## 1 Introduction

Approximate expressions for generalization errors for finite dimensional statistical data models can be often obtained in the large data limit using asymptotic expansions. Such methods can yield approximate relations for empirical and true errors which can be used to assess the quality of the trained model see e.g. [1]. Unfortunately, such an approximation scheme does not seem to be easily applicable to popular *non-parametric* models like Gaussian process (GP) models and Support Vector Machines (SVMs). We apply the replica approach of statistical physics to asses the average case learning performance of these kernel machines. So far, the tools of statistical physics have been successfully applied to a variety of learning problems [2]. However, this elegant method suffers from the drawback that data averages can be performed exactly only under very idealistic assumptions on the data distribution in the "thermodynamic" limit of infinite data space dimension. We try to overcome these limitations by combining the replica method with a variational approximation. For Bayesian models, our method allows us to express useful data averaged a-posteriori expectations by means of an approximate measure. The derivation of this measure requires no assumptions about the data density and no assumptions about the input dimension.

The main focus of this article is Gaussian process regression where we demonstrate the various strengths of the presented method. It solves some of the problems stated at the end of our previous NIPS paper [3] which was based on a simpler somewhat unmotivated truncation of a cumulant expansion. For Gaussian process models we show that our method does not only give explicit approximations for generalization errors but also of their sample fluctuations. Furthermore, we show how to compute corrections to our theory and demonstrate the possibility of deriving approximate universal relations between average empirical and true errors which might be of practical interest.

An earlier version of our approach, which was still restricted to the assumption of idealized data distributions appeared in [4].

## 2 Setup and Notation

We assume that a class of elementary predictors (neural networks, regressors etc.) is given by functions $f(x)$. In a Bayesian formulation, we have a prior distribution over this class of functions $f$. Assuming that a set of observations $y_1, \ldots, y_m$ is conditionally independent given inputs $x_1, \ldots, x_m$, we assign a likelihood term of the form $\exp(-\beta h(f(x_i), y_i))$ to each observation. Posterior expectations (denoted by angular brackets) of any functional $F\{f\}$ are expressed in the form

$$\langle F\{f\} \rangle = \frac{1}{Z_m} E\left(F\{f\} \exp\left(-\beta \sum_{i=1}^{m} h(f(x_i), y_i)\right)\right) \tag{1}$$

where the partition function $Z_m$ normalizes the posterior and $E$ denotes the expectation with respect to the prior. We are interested in computing averages $[\langle F\{f\}\rangle]_D$ of posterior expectations over different drawings of training data sets $D = \{(x_1, y_1), \ldots, (x_m, y_m)\}$ were all data examples are independently generated from the same distribution. In the next section we will show how to derive a measure which enables us to compute analytically approximate combined data and posterior averages.

## 3 A Grand-Canonical Approach

We utilize the statistical mechanics approach to the analysis of learning. Our aim is to compute the so-called averaged "free energy" $[-\ln Z_m]_D$ which serves as a generating function for suitable data averages of posterior expectations. The partition function $Z_m$ is

$$Z_m = E \exp\left(-\beta \sum_{i=1}^{m} h(f(x_i), y_i)\right). \tag{2}$$

To perform the average $[\ln Z_m]_D$ we use the replica trick $[\ln Z_m]_D = \lim_{n \to 0} \frac{\partial \ln[Z_m^n]_D}{\partial n}$, where $[Z_m^n]_D$ is computed for integer $n$ and the continuation is performed at the end [5]. We obtain

$$Z_n(m) \doteq [Z_m^n]_D = E_n\left(\left[\exp\left(-\beta \sum_{a=1}^{n} h(f_a(x), y)\right)\right]_{(x,y)}\right)^m, \tag{3}$$

where $E_n$ denotes the expectation over the replicated prior measure.

Eq.(3) can be transformed into a simpler form by introducing the "grand canonical" partition function $\Xi_n(\mu)$

$$\Xi_n(\mu) \doteq \sum_{m=0}^{\infty} \frac{e^{\mu m}}{m!} Z_n(m) = E_n \exp(-H_n) \tag{4}$$

with the Hamiltonian

$$H_n = -e^{\mu}\left[\exp\left(-\beta \sum_{a=1}^{n} h(f_a(x), y)\right)\right]_{(x,y)}. \tag{5}$$

The density $e^{-H_n}$ evaluates all $n$ replicas $f_1, \ldots, f_n$ of the predictor at the *same* data point $(x, y)$ and the expectation $[\cdots]_{(x,y)}$ is taken with respect to the true data density $p(x, y)$.

The "grand canonical" partition function $\Xi_n(\mu)$ represents a "poissonized" version of the original model with fluctuating number of examples. The "chemical potential" $\mu$ determines the expected value of $m = \frac{\partial \ln \Xi_n(\mu)}{\partial \mu}$ which yields simply $\mu = \ln m$ for $n \to 0$. For sufficiently large $m$, we can replace the sum in Eq. (4) by its dominating term

$$\ln Z_n(m) \approx \ln \Xi_n(\mu) + m(\ln m - 1) - m\mu. \tag{6}$$

thereby neglecting relative fluctuations. We recover the original (canonical) free energy as
$-\frac{\partial \ln Z_n(m)}{\partial n} \approx -\frac{\partial \ln \Xi_n(\ln m)}{\partial n}$.

# 4 Variational Approximation

For most interesting cases, the partition function $\Xi_n(\mu)$ can not be computed in closed form for given $n$. Hence, we use a variational approach to approximate $H_n$ by a different tractable Hamiltonian $H_n^0$. It is easy to write down the first terms in an expansion of the "grand canonical" free energy with respect to the difference $H_n - H_n^0$

$$-\ln \Xi_n(\mu) = -\ln E_n e^{-H_n^0} + \langle H_n - H_n^0 \rangle_0 - \frac{1}{2}\left(\langle (H_n - H_n^0)^2 \rangle_0 - \langle H_n - H_n^0 \rangle_0^2\right) \pm \ldots. \quad (7)$$

The brackets $\langle \ldots \rangle_0$ denote averages with respect to the effective measure which is induced by the prior and $e^{-H_n^0}$ and acts in the space of replicated variables. As is well known, the first two leading terms in Eq.(7) present an upper bound [6] to $-\ln \Xi_n(\mu)$. Although differentiating the bound with respect to $n$ will usually not preserve the inequality, we still expect [1] that an optimization with respect to $H_n^0$ is a sensible thing to do [7].

## 4.1 Variational Equations

The grand-canonical ensemble was chosen such that Eq.(5) can be rewritten as an integral over a local quantity in the input variable $x$, i.e. in the form $H_n = -m \int dx \, \mathcal{H}(x, \{f_a(x)\})$ with

$$\mathcal{H}(x, \{f_a(x)\}) = \int dy \, p(y, x) \exp\left(-\beta \sum_{a=1}^{n} h(f_a(x), y)\right). \quad (8)$$

We will now specialize to Gaussian priors over $f$, for which a local quadratic expression

$$H_n^0 = \int dx \sum_{ab} \frac{1}{2}\eta_{ab}(x) f_a(x) f_b(x) + \sum_a \hat{r}_a(x) f_a(x) \quad (9)$$

is a suitable trial Hamiltonian, leading to Gaussian averages $\langle \ldots \rangle_0$. The functions $\eta_{ab}(x)$ and $\hat{r}_a(x)$ are variational parameters to be optimized. It is important to have an explicit dependence on the input variable $x$ in order to take a non uniform input density into account.

To perform the variation of the first two terms in Eq.(7) we note that the locality of Eq.(8) makes the "variational free energy" $-\ln E_n \exp(-H_n^0) + \langle H_n - H_n^0 \rangle_0$ an explicit function of the first two local moments

$$K_{ab}(x) \doteq \langle f_a(x) f_b(x) \rangle_0 \qquad r_a(x) \doteq \langle f_a(x) \rangle_0. \quad (10)$$

Hence, a straightforward variation yields

$$m \frac{d\langle \mathcal{H}(x, f(x)) \rangle_0}{dK_{ab}(x)} = \eta_{ab}(x) \qquad m \frac{d\langle \mathcal{H}(x, f(x)) \rangle_0}{d\, r_a(x)} = \hat{r}_a(x). \quad (11)$$

To extend the variational solutions to non-integer values of $n$, we assume that for all $a$ the optimal parameters are replica symmetric, ie. $\hat{r}_a(x) = \hat{r}(x)$ as well as $\eta_{ab}(x) = \eta(x)$ for $a \neq b$ and $\eta_{aa}(x) = \eta_0(x)$. We also use a corresponding notation for $K_{ab}(x)$ and $r_a(x)$.

## 4.2 Interpretation of $H_n^0$

Note, that our approach is not equivalent to a variational approximation of the original posterior. In contrast, $H_0$ contains the full information of the statistics of the training data. We can use the distribution induced by the prior and $e^{-H_n^0}$ in order to compute approximate combined data and posterior averages. As an example, we first consider the expected local *posterior variance* $\sigma^2(x) \doteq [\langle f^2(x)\rangle - \langle f(x)\rangle^2]_D$. Following the algebra of the replica method (see [5]) this is approximated within the variational replica approach as

$$\sigma^2(x) = \lim_{n\to 0} \left(\langle f_a^2(x)\rangle_0 - \langle f_a(x)f_b(x)\rangle_0 \right) = K_0(x) - K(x) . \tag{12}$$

Second, we consider the noisy local *mean square prediction error* of the posterior mean predictor $\hat{f}(x) = \langle f(x)\rangle$ which is given by $\varepsilon(x,y) = [(\hat{f}(x) - y)^2]_D$. In this case

$$\begin{aligned}
\varepsilon(x,y) &= \lim_{n\to 0} \left(\langle f_a(x)f_b(x)\rangle_0 + y^2 - 2y\langle f_a(x)\rangle_0 \right) \\
&= K(x) - r^2(x) + (r(x) - y)^2.
\end{aligned} \tag{13}$$

We can also calculate *fluctuations* with respect to the data average, for example

$$[(\hat{f}(x) - y)^2(\hat{f}(x') - y')^2]_D = \lim_{n\to 0} \left\langle \prod_{a=1,2} (f_a(x) - y) \prod_{b=3,4} (f_b(x') - y') \right\rangle_0 . \tag{14}$$

## 5 Regression with Gaussian Processes

This statistical model assumes that data are generated as $y_i = f(x_i) + \xi_i$, where $\xi$ is Gaussian white noise with variance $\beta^{-1}$. The prior over functions has zero mean and covariance $C(x,x') = E[f(x)f(x')]$. Hence, we have $h(f,y) = \frac{1}{2}(y - f(x))^2$. Using the definitions Eqs.(12,13), we get

$$\frac{\partial \langle \mathcal{H}(x, \{f_a(x)\})\rangle_0}{\partial n} = -\frac{p(x)}{2} \left\{ \ln(1 + \beta\sigma^2(x)) + \frac{\int dy\, p(y|x)\varepsilon(x,y)}{\beta^{-1} + \sigma^2(x)} \right\} \tag{15}$$

which yields the set of variational equations (11). They become particularly easy when the regression model uses a translationally invariant kernel $C(x - x')$ and the input distribution is homogeneous in a finite interval. The variational equations (11) can then be solved in terms of the eigenvalues of the Gaussian process kernel.

[8, 9] studied learning curves for Gaussian process regression which are not only averaged over the data but also over the data generating process $f_*$ using a Gaussian process prior on $f_*$. Applying these averages to our theory and adapting the notation of [9] simply replaces in Eq.(15) the term $\int dy\, p(y|x)\varepsilon(x,y)$ by $\epsilon(x) + \sigma_*^2$ while $\sigma^2(x) \equiv \hat{\epsilon}(x)$.

### 5.1 Learning Curves and Fluctuations

Practical situations differ from this "typical case" analysis. The data generating process is unknown but assumed to be *fixed*. The resulting learning curve is then conditioned on this particular "teacher" $f_*$. The left panel of Fig.1 shows an example. Displayed are the mean square prediction error $\varepsilon$ (circle and solid line) and its sample fluctuations (error bars) $\Delta\varepsilon$ with respect to the data average (cross and broken line). The target $f_*$ was a random but fixed realization from a Gaussian process prior with a periodic Radial Basis Function kernel $C(x,x') = \sum_k \exp(-(x - x' - k)^2/2l^2), l = 0.1$. We keep the example simple, e.g the Gaussian process regression model used the same kernel and noise $\beta^{-1} = \sigma_*^2 = 0.01$. The inputs are one dimensional, independent and uniformly distributed $x \in [0,1]$. Symbols represent simulation data. A typical property of our theory (lines) is that it becomes very accurate for sufficiently large number of example data.

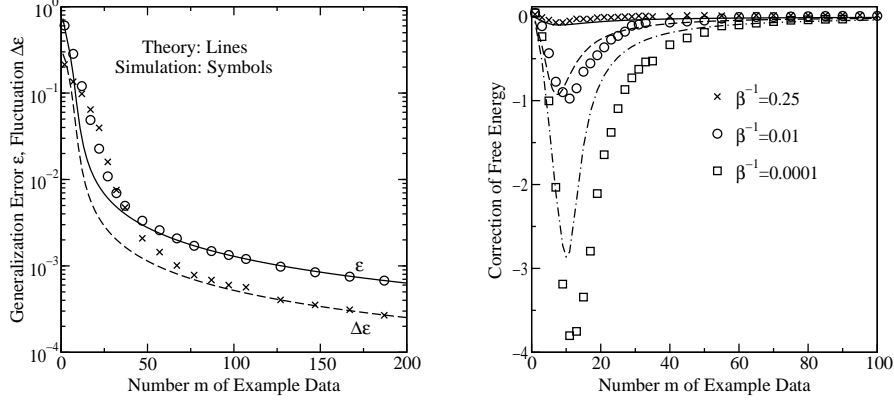

Figure 1: Gaussian process regression using a periodic Radial Basis Function kernel, input dimension d=1, $x \in [0, 1]$, and homogeneous input density. *Left:* Generalization error $\varepsilon$ and fluctuations $\Delta\varepsilon$ for data noise $\sigma_*^2 = \beta^{-1} = 0.01$. *Right:* Correction of the free energy. Symbols: We subtracted the first two contributions to Eq.(7) from the true value of the free energy. The latter was obtained by simulations. Lines show the third contribution of Eq.(7). The value of the noise variance $\beta^{-1}$ decreases from top to bottom. All y-data was set equal to zero.

## 5.2 Corrections to the Variational Approximation

It is a strength of our method that the quality of the variational approximation Eq.(7) can be characterized and systematically improved. In this paper, we restrict ourself to a characterization and consider the case where all $y$-data is set equal to zero. Since the posterior variance $\sigma^2(x)$ is independent of the data this is still an interesting model from which the posterior variance can be estimated. We consider the third term in the expansion to the free energy Eq.(7). It is a correction to the variational free energy and evaluates to

$$- \lim_{n \to 0} \frac{\partial}{\partial n} \frac{1}{2} \left( \langle (H_n - H_n^0)^2 \rangle_0 - \langle H_n - H_n^0 \rangle_0^2 \right) = -\frac{1}{4} [\eta_0(x)\eta_0(x')\hat{C}^2(x, x')]_{x', x}$$

$$+ \frac{m^2}{4} \left[ \ln \left( 1 - \frac{\hat{C}^2(x, x')}{(\sigma^2(x) + \beta^{-1})^2} \right) \right]_{x', x} + \frac{m}{2} \left[ \frac{\eta_0(x')\hat{C}^2(x, x')}{(\sigma^2(x) + \beta^{-1})} \right]_{x', x} \quad (16)$$

with $\hat{C}(x, x') = \lim_{n \to 0} \langle f_a(x)f_a(x') - f_a(x)f_b(x') \rangle_0$. Eq.(16) is shown by lines in the right panel of Fig.1 for different values of the model noise $\beta^{-1}$. We considered a homogeneous input density, the input dimension is one and the regression model uses a periodic RBF kernel. The symbols in Fig.1 show the difference between the true value of the free energy which is obtained by simulations and the first two terms of Eq.(7). The correction term is found to be qualitatively accurate and emphasizes a discrepancy between free energy and the first two terms of the expansion Eq.(7) for a medium amount of example data. The calculated learning curves inherit this behaviour.

## 5.3 Universal Relations

We can relate the training error $\varepsilon_T$ and the empirical posterior variance $\sigma_T^2$

$$\varepsilon_T \doteq \frac{1}{m} \left[ \sum_{i=1}^{m} \left( \hat{f}(x_i) - y_i \right)^2 \right]_D \quad ; \quad \sigma_T^2 \doteq \frac{1}{m} \left[ \sum_{i=1}^{m} \sigma^2(x_i) \right]_D \quad (17)$$

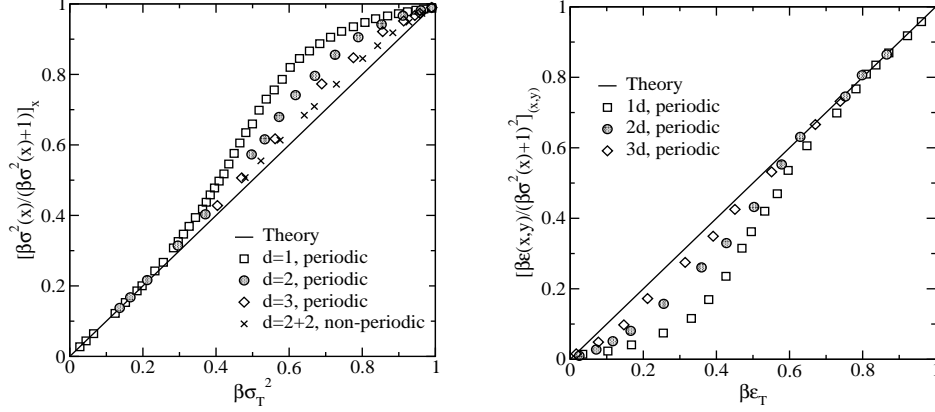

Figure 2: Illustration of relation Eq.(19) (*left*) and Eq.(20) (*right*). All error measures are scaled with $\beta$. Symbols show simulation results for Radial Basis Function (RBF) regression and a homogeneous input distribution in $d = 1, 2, 3$ dimensions (square, circle, diamond). The RBF kernel was periodic. Additionally, the left figure shows an example were the inputs lie on a quasi two-dimensional manifold which is embedded in $d = 4$ dimensions (cross). In this case the RBF kernel was non-periodic.

to the free energy $\frac{d}{d\beta}[-\ln Z_m]_D = \frac{m}{2}(\varepsilon_T + \sigma_T^2)$. Using Eqs.(6,7) and the stationarity of the grand-canonical free energy with respect to the variational parameters we obtain the following relation

$$\frac{d}{d\beta}[-\ln Z_m]_D \approx -m\frac{\partial}{\partial\beta} \int dx \frac{\partial\langle\mathcal{H}(x,\{f_a(x)\})\rangle_0}{\partial n}. \tag{18}$$

We use the fact that the posterior variance is independent of the $y$-data and simply estimate it from the model where all $y$-data is set equal to zero. In this case, Eq.(18) yields

$$\sigma_T^2 = \int dx\, p(x)\, \frac{\sigma^2(x)}{1 + \beta\sigma^2(x)} \tag{19}$$

which relates the empirical posterior variance $\sigma_T^2$ to the local posterior variance $\sigma^2(x)$ at test inputs $x$. Similarly, we can derive an expression for the training error $\varepsilon_T$ by using Eqs.(15,18) in combination with Eq.(19)

$$\varepsilon_T = \int dx\, dy\, p(x,y)\, \frac{\varepsilon(x,y)}{(1 + \beta\sigma^2(x))^2}. \tag{20}$$

It is interesting to note, that the relations (19,20) contain no assumptions about the data generating process. They hold in general for Gaussian process models with a Gaussian likelihood. An illustration of Eqs.(19,20) is given by Fig.2 for the example of Gaussian process regression with a Radial Basis Function kernel. In the left panel of Fig.2, learning starts in the upper right corner as the rescaled empirical posterior variance $\beta\sigma_T^2$ is initially one and decreases with increasing number of example data. For the right panel of Fig.2, learning starts in the lower left corner. The rescaled training error $\beta\varepsilon_T$ on the noisy data set is initially zero and increases to one with increasing number of example data. The theory (line) holds for a sufficiently large number of example data and its accuracy increases with the input dimension. Eqs.(19,20) can also be tested on *real* data. For common benchmark sets such as Abalone and Boston Housing data we find that Eqs.(19,20) hold well even for small and medium sizes of the training data set.

# 6   Outlook

One may question if our approximate universal relations are of any practical use as, for example, the relation between training error and generalization error involves also the unknown posterior variance $\sigma^2(x)$. Nevertheless, this relation could be useful for cases, where a large number of *data inputs without output labels* are available. Since for regression, the posterior variance is independent of the output labels, we could use these extra input points to estimate $\sigma^2(x)$.

The application of our technique to more complicated models is possible and technically more involved. For example, replacing $e^{-\beta h}$ by $\Theta(yf(x)-1)$ in Eq.(1) and further rescaling the kernel $C(x,x')=K(x,x')/\gamma$ of the Gaussian process prior gives a model for hard margin *Support Vector Machine* Classification with SVM kernel $K(x,x')$. The condition of maximum margin classification will be ensured by the limes $\gamma \to \infty$.

Of particular interest is the computation of empirical estimators that can be used in practice for model selection as well as the calculation of fluctuations (error bars) for such estimators. A prominent example is an efficient approximate leave-one-out estimator for SVMs.

Work on these issues is in progress.

## Acknowledgement

We would like to thank Peter Sollich for may inspiring discussions. The work was supported by EPSRC grant GR/M81601.

## Footnotes

[1]Guided by the success of the method in physical applications, for instance in polymer physics.

## References

[1] N. Murata, S. Yoshizawa, S. Amari, *IEEE Transactions on Neural Networks* **5**, p. 865-872, (1994).

[2] A. Engel, C. Van den Broeck, *Statistical Mechanics of Learning*, Cambridge University Press (2001).

[3] D. Malzahn, M. Opper, *Neural Information Processing Systems 13*, p. 273, T. K. Leen, T. G. Dietterich and V. Tresp, eds., MIT Press, Cambridge MA (2001).

[4] D. Malzahn, M. Opper, Lecture Notes in Computer Science 2130, p. 271, G. Dorffner, H. Bischof and K. Hornik, eds., Springer, Berlin (2001).

[5] M. Mézard, G. Parisi, M. Virasoro, *Spin Glass Theory and Beyond*, World Scientific, Singapore, (1987).

[6] R. P. Feynman and A. R. Hibbs, Quantum mechanics and path integrals, Mc Graw-Hill Inc., (1965).

[7] T. Garel, H. Orland, Europhys. Lett. 6, p. 307 (1988).

[8] P. Sollich, *Neural Information Processing Systems 11*, p. 344, M. S. Kearns, S. A. Solla and D. A. Cohn, eds., MIT Press, Cambridge MA (1999).

[9] P. Sollich, *Neural Information Processing Systems 14*, T. G. Dietterich, S. Becker, Z. Ghahramani, eds., MIT Press (2002).
